# Estimating Dependency Structure as a Hidden Variable

**Marina Meilă and Michael I. Jordan**
{mmp, jordan}@ai.mit.edu
Center for Biological & Computational Learning
Massachusetts Institute of Technology
45 Carleton St. E25-201
Cambridge, MA 02142

## Abstract

This paper introduces a probability model, the *mixture of trees* that can account for sparse, dynamically changing dependence relationships. We present a family of efficient algorithms that use EM and the Minimum Spanning Tree algorithm to find the ML and MAP mixture of trees for a variety of priors, including the Dirichlet and the MDL priors.

## 1 INTRODUCTION

A fundamental feature of a good model is the ability to uncover and exploit independencies in the data it is presented with. For many commonly used models, such as neural nets and belief networks, the dependency structure encoded in the model is fixed, in the sense that it is not allowed to vary depending on actual values of the variables or with the current case. However, dependency structures that are conditional on values of variables abound in the world around us. Consider for example bitmaps of handwritten digits. They obviously contain many dependencies between pixels; however, the pattern of these dependencies will vary across digits. Imagine a medical database recording the body weight and other data for each patient. The body weight could be a function of age and height for a healthy person, but it would depend on other conditions if the patient suffered from a disease or was an athlete.

Models that are able to represent data conditioned dependencies are decision trees and mixture models, including the soft counterpart of the decision tree, the mixture of experts. Decision trees however can only represent certain patterns of dependecy, and in particular are designed to represent a set of conditional probability tables and not a joint probability distribution. Mixtures are more flexible and the rest of this paper will be focusing on one special case called the *mixtures of trees*.

We will consider domains where the observed variables are related by pairwise dependencies only and these dependencies are sparse enough to contain no cycles. Therefore they can

be represented graphically as a *tree*. The structure of the dependencies may vary from one instance to the next. We index the set of possible dependecy structures by a discrete *structure variable z* (that can be observed or hidden) thereby obtaining a *mixture*.

In the framework of graphical probability models, tree distributions enjoy many properties that make them attractive as modelling tools: they have a flexible topology, are intuitively appealing, sampling and computing likelihoods are linear time, simple efficient algorithms for marginalizing and conditioning ($\mathcal{O}(|V|^2)$ or less) exist. Fitting the best tree to a given distribution can be done exactly and efficiently (Chow and Liu, 1968). Trees can capture simple pairwise interactions between variables but they can prove insufficient for more complex distributions. Mixtures of trees enjoy most of the computational advantages of trees and, in addition, they are universal approximators over the space of all distributions. Therefore, they are fit for domains where the dependency patterns become tree like when a possibly hidden variable is instantiated.

Mixture models have been extensively used in the statistics and neural network literature. Of relevance to the present work are the mixtures of Gaussians, whose distribution space, in the case of continuous variables overlaps with the space of mixtures of trees. Work on fitting a tree to a distribution in a Maximum-Likelihood (ML) framework has been pioneered by (Chow and Liu, 1968) and was extended to polytrees by (Pearl, 1988) and to mixtures of trees with observed structure variable by (Geiger, 1992; Friedman and Goldszmidt, 1996). Mixtures of factorial distributions were studied by (Kontkanen et al., 1996) whereas (Thiesson et al., 1997) discusses mixtures of general belief nets. Multinets (Geiger, 1996) which are essentially mixtures of Bayes nets include mixtures of trees as a special case. It is however worth studying mixtures of trees separately for their special computational advantages.

This work presents efficient algorithms for learning mixture of trees models with unknown or hidden structure variable. The following section introduces the model; section 3 develops the basic algorithm for its estimation from data in the ML framework. Section 4 discusses the introduction of priors over mixtures of trees models and presents several realistic factorized priors for which the MAP estimate can be computed by a modified versions of the basic algorithm. The properties of the model are verified by simulation in section 5 and section 6 concludes the paper.

## 2 THE MIXTURE OF TREES MODEL

In this section we will introduce the mixture of trees model and the notation that will be used throughout the paper. Let $V$ denote the set of variables of interest. According to the graphical model paradigm, each variable is viewed as a vertex of a graph. Let $r_v$ denote the number of values of variable $v \in V$, $x_v$ a particular value of V, $x_A$ an assignment to the variables in the subset $A$ of $V$. To simplify notation $x_V$ will be denoted by $x$.

We use trees as graphical representations for families of probability distributions over $V$ that satisfy a common set of independence relationships encoded in the tree topology. In this representation, an edge of the tree shows a direct dependence, or, more precisely, the absence of an edge between two variables signifies that they are independent, conditioned on all the other variables in $V$. We shall call a graph that has no cycles a *tree*[1] and shall denote by $E$ the set of its (undirected) edges. A probability distribution $T$ that is conformal with the tree $(V, E)$ is a distribution that can be factorized as:

$$T(x) = \frac{\prod_{(u,v)\in E} T_{uv}(x_u, x_v)}{\prod_{v\in V} T_v(x_v)^{\deg v - 1}} \qquad (1)$$

Here deg $v$ denotes the *degree* of $v$, e.g. the number of edges incident to node $v \in V$. The

factors $T_{uv}$ and $T_v$ are the marginal distributions under $T$:

$$T_{uv}(x_u, x_v) = \sum_{x_{V-\{u,v\}}} T(x_u, x_v, x_{V-\{u,v\}}), \quad T_v(x_v) = \sum_{x_{V-\{v\}}} T(x_v, x_{V-\{v\}}). \quad (2)$$

The distribution itself will be called a tree when no confusion is possible. Note that a tree distribution has for each edge $(u, v) \in E$ a factor depending on $x_u, x_v$ onlyl If the tree is connected, e.g. it *spans* all the nodes in $V$, it is often called a *spanning tree*.

An equivalent representation for $T$ in terms of conditional probabilities is

$$T(x) = \prod_{v \in V} T_{v|\text{pa}(v)}(x_v | x_{\text{pa}(v)}) \quad (3)$$

The form (3) can be obtained from (1) by choosing an arbitrary root in each connected component and recursively substituting $\frac{T_{v\text{pa}(v)}}{T_v}$ by $T_{v|\text{pa}(v)}$ starting from the root. $\text{pa}(v)$ represents the parent of $v$ in the thus directed tree or the empty set if $v$ is the root of a connected component. The directed tree representation has the advantage of having independent parameters. The total number of free parameters in either representation is $\sum_{(u,v)\in E_T} r_u r_v - \sum_{v \in V} (\deg v - 1) r_v$.

Now we define a mixture of trees to be a distribution of the form

$$Q(x) = \sum_{k=1}^{m} \lambda_k T^k(x); \qquad \lambda_k \geq 0, \, k = 1, \ldots, m; \qquad \sum_{k=1}^{m} \lambda_k = 1. \quad (4)$$

From the graphical models perspective, a mixture of trees can be viewed as a containing an unobserved choice variable $z$, taking value $k \in \{1, \ldots m\}$ with probability $\lambda_k$. Conditioned on the value of $z$ the distribution of the visible variables $x$ is a tree. The $m$ trees may have different structures and different parameters. Note that because of the structure variable, a mixture of trees is not properly a belief network, but most of the results here owe to the belief network perspective.

# 3  THE BASIC ALGORITHM: ML FITTING OF MIXTURES OF TREES

This section will show how a mixture of trees can be fit to an observed dataset in the Maximum Likelihood paradigm via the EM algorithm (Dempster et al., 1977). The observations are denoted by $\{x^1, x^2, \ldots, x^N\}$; the corresponding values of the structure variable are $\{z^i, i = 1, \ldots N\}$.

Following a usual EM procedure for mixtures, the Expectation (E) step consists in estimating the posterior probability of each tree to generate datapoint $x^i$

$$Pr[z^i = k | x^{1,\ldots,N}, model] = \gamma_k(i) = \frac{\lambda_k T^k(x^i)}{\sum_{k'} \lambda_{k'} T^{k'}(x^i)} \quad (5)$$

Then the expected complete log-likelihood to be maximized by the M step of the algorithm is

$$E[l_c | x^{1,\ldots N}, model] = \sum_{k=1}^{m} \Gamma_k [\log \lambda_k + \sum_{i=1}^{N} P^k(x^i) \log T^k(x^i)] \quad (6)$$

$$\Gamma_k = \sum_{i=1}^{N} \gamma_k(x^i), \qquad k = 1, \ldots m \text{ and } P^k(x^i) = \gamma_k(i)/\Gamma_k. \quad (7)$$

The maximizing values for the parameters $\lambda$ are $\lambda_k^{new} = \Gamma_k/N$. To obtain the new distributions $T^k$, we have to maximize for each $k$ the expression that is the negative of the

Figure 1: **The Basic Algorithm: ML Fitting of a Mixture of Trees**
**Input:** Dataset $\{x^1, \ldots x^N\}$
   Initial model $m$, $T^k$, $\lambda^k$, $k = 1, \ldots m$
   Procedure MST( weights ) that fits a maximum weight spanning tree over $V$
Iterate until convergence
 **E step:**   compute $\gamma_k^i$, $P^k(x^i)$ for $k = 1, \ldots m$, $i = 1, \ldots N$ by (5), (7)
 **M step:**
   **M1.**   $\lambda_k \leftarrow \Gamma_k/N$, $k = 1, \ldots m$
   **M2.**   compute marginals $P_v^k$, $P_{uv}^k$, $u, v \in V$, $k = 1, \ldots m$
   **M3.**   compute mutual information $I_{uv}^k u, v \in V$, $k = 1, \ldots m$
   **M4.**   call MST($\{ I_{uv}^k \}$) to generate $E_{T^k}$ for $k = 1, \ldots m$
   **M5.**   $T_{uv}^k \leftarrow P_{uv}^k, ; T_v^k \leftarrow P_v^k$ for $(u, v) \in E_{T^k}$, $k = 1, \ldots m$

crossentropy between $P^k$ and $T^k$.

$$\sum_{i=1}^N P^k(x^i) \log T^k(x^i) \tag{8}$$

This problem can be solved exactly as shown in (Chow and Liu, 1968). Here we will give a brief description of the procedure. First, one has to compute the mutual information between each pair of variables in $V$ under the target distribution $P$

$$I_{uv} = I_{vu} = \sum_{x_u x_v} P_{uv}(x_u, x_v) \log \frac{P_{uv}(x_u, x_v)}{P_u(x_u)P_v(x_v)}, \quad u, v \in V, u \neq v. \tag{9}$$

Second, the optimal tree structure is found by a *Maximum Spanning Tree* (MST) algorithm using $I_{uv}$ as the weight for edge $(u, v), \forall u, v \in V$. Once the tree is found, its marginals $T_{uv}$ (or $T_{u|v}$), $(u, v) \in E_T$ are exactly equal to the corresponding marginals $P_{uv}$ of the target distribution $P$. They are already computed as an intermediate step in the computation of the mutual informations $I_{uv}$ (9).

In our case, the target distribution for $T^k$ is represented by the posterior sample distribution $P^k$. Note that although each tree fit to $P^k$ is optimal, for the encompassing problem of fitting a mixture of trees to a sample distribution only a local optimum is guaranteed to be reached. The algorithm is summarized in figure 1.

This procedure is based on one important assumption that should be made explicit now. It is the **Parameter independence assumption**: *The distribution $T_{v|pa(v)}^k$ for any $k$, $v$ and value of $pa(v)$ is a multinomial with $r_v - 1$ free parameters that are independent of any other parameters of the mixture.*

It is possible to constrain the $m$ trees to share the same structure, thus constructing a truly Bayesian network. To achieve this, it is sufficient to replace the weights in step **M4** by $\sum_k I_{uv}^k$ and run the MST algorithm only once to obtain the common structure $E_T$. The tree stuctures obtained by the basic algorithm are connected. The following section will give reasons and ways to obtain disconnected tree structures.

## 4   MAP MIXTURES OF TREES

In this section we extend the basic algorithm to the problem of finding the Maximum a Posteriori (MAP) probability mixture of trees for a given dataset. In other words, we will consider a nonuniform prior $P[model]$ and will be searching for the mixture of trees that maximizes

$$\log P[model|x^{1,\ldots N}] = \log P[x^{1,\ldots N}|model] + \log P[model] + \text{constant}. \tag{10}$$

**Factorized priors** The present maximization problem differs from the ML problem solved in the previous section only by the addition of the term $\log P[model]$. We can as well

approach it from the EM point of view, by iteratively maximizing

$$E\left[\log P[model|x^{1,...N}, z^{1,...N}]\right] = E[l_c(x^{1,...N}, z^{1,...N}|model)] + \log P[model] \quad (11)$$

It is easy to see that the added term does not have any influence on the E step, which will proceed exactly as before. However, in the M step, we must be able to successfully maximize the r.h.s. of (11). Therefore, we look for priors of the form

$$P[model] = P[\lambda_{1,...m}] \prod_{k=1}^{m} P[T_k] \quad (12)$$

This class of priors is in agreement with the parameter independence assumption and includes the conjugate prior for the multinomial distribution which is the Dirichlet prior. A Dirichlet prior over a tree can be represented as a table of fictitious marginal probabilities $P_{uv}'^k$ for each pair $u, v$ of variables plus an *equivalent sample size* $N'$ that gives the strength of the prior (Heckerman et al., 1995). However, for Dirichlet priors, the maximization over tree structures (corresponding to step **M4**) can only be performed iteratively (Meilă et al., 1997).

**MDL (Minimum Description Length) priors** are less informative priors. They attempt to balance the number of parameters that are estimated with the amount of data available, usually by introducing a penalty on model complexity. For the experiments in section 5 we used *edge pruning*. More smoothing methods are presented in (Meilă et al., 1997). To penalize the number of parameters in each component we introduce a prior that penalizes each edge that is added to a tree, thus encouraging the algorithm to produce disconnected trees. The edge pruning prior is $P[T] \propto \exp\left[-\beta \sum_{uv \in E_T} \Delta_{uv}\right]$. We choose a uniform penalty $\Delta_{uv} = 1$. Another possible choice is $\Delta_{uv} = (r_u - 1)(r_v - 1)$ which is the number of parameters introduced by the presence of edge $(u, v)$ w.r.t. a factorized distribution. Using this prior is equivalent to maximizing the following expression in step **M4** of the Basic Algorithm (the index $k$ being dropped for simplicity)

$$\operatorname*{argmax}_{E_T} \sum_{uv \in E_T} \max[0, \Gamma I_{uv} - \beta \Delta_{uv}] = \operatorname*{argmax}_{E_T} \sum_{uv \in E_T} W_{uv} \quad (13)$$

## 5  EMPIRICAL RESULTS

We have tested our model and algorithms for their ability to retrieve the dependency structure in the data, as classifiers and as density estimators.

For the first objective, we sampled 30,000 datapoints from a mixture of 5 trees over 30 variables with $r_v = 4$ for all vertices. All the other parameters of the generating model and the initial points for the algorithm were picked at random. The results on retrieving the original trees were excellent: out of 10 trials, the algorithm failed to retrieve correctly only 1 tree in 1 trial. This bad result can be accounted for by sampling noise. The tree that wasn't recovered had a $\lambda$ of only 0.02. The difference between the log likelihood of the samples of the generating tree and the approximating tree was 0.41 bits per example.

For classification, we investigated the performance of mixtures of trees on a the Australian Credit dataset from the UCI repository[2]. The data set has 690 instances of 14-dimensional attribute vectors. Nine attributes are discrete ( $2 - 14$ values) and 5 are continuous. The class variable has 6 values. The continuous variables were discretized in $3 - 5$ uniform bins each. We tested mixtures with different values for $m$ and for the edge pruning parameter $\beta$. For comparison we tried also mixtures of factorial distributions of different sizes. One tenth of the data, picked randomly at each trial, was used for testing and the rest for training. In the training phase, we learned a MT model of the joint distribution of all the 15 variables.

Figure 2: Performance of different algorithms on the Australian Credit dataset. – is mixture of trees with $\beta = 10$, - - is mixture of trees with $beta = 1/m$, --- is mixture of factorial distributions.

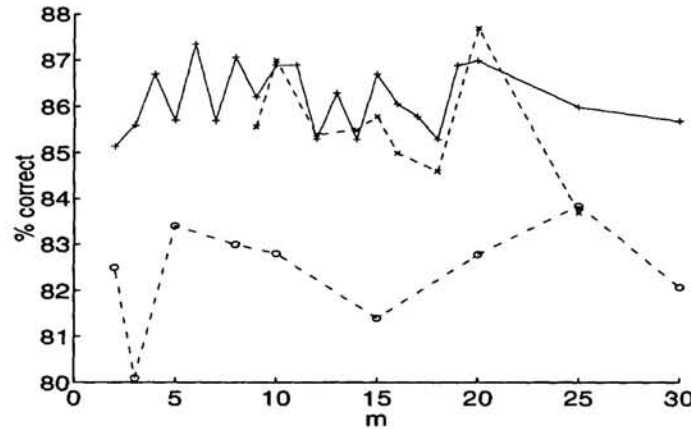

Table 1: a) Mixture of trees compression rates [$\log l_{test}/N_{test}$]. b) Compression rates (bits/digit) for the single digit (Digit) and double digit (Pairs) datasets. MST is mixtures of trees, MF is a mixture of factorial distributions, BR is base rate model, H-WS is Helmholtz Machine trained with the wake-sleep algorithm (Frey et al., 1996), H-MF is Helmholtz Machine trained with the Mean Field approximation, FV is a fully visible bayes net. (*=best)

(a)

| $m$ | Digits | Pairs |
|---|---|---|
| 16 | *34.72 | 79.25 |
| 32 | 34.48 | *78.99 |
| 64 | 34.84 | 79.70 |
| 128 | 34.88 | 81.26 |

(b)

| Algorithm | Digits | Pairs |
|---|---|---|
| gzip | 44.3 | 89.2 |
| BR | 59.2 | 118.4 |
| MF | 37.5 | 92.7 |
| H-MF | 39.5 | 80.7 |
| H-WS | 39.1 | 80.4 |
| FV | 35.9 | *72.9 |
| **MT** | *34.7 | 79.0 |

In the testing phase, the output of our classifier was chosen to be the class value with the largest posterior probability given the inputs. Figure 2 shows that the results obtained for mixtures of trees are superior to those obtained for mixtures of factorial distributions.For comparison, correct classification rates obtained and cited in (Kontkanen et al., 1996) on training/test sets of the same size are: 87.2next best model (a decision tree called Cal50).

We also tested the basic algorithm as a density estimator by running it on a subset of binary vector representations of handwritten digits and measuring the compression rate. One dataset contained images of single digits in 64 dimensions, the second contained 128 dimensional vectors representing randomly paired digit images. The training, validation and test set contained 6000, 2000, and 5000 exemplars respectively. The data sets, the training conditions and the algorithms we compared with are described in (Frey et al., 1996). We tried mixtures of 16, 32, 64 and 128 trees, fitted by the basic algorithm. The results (shown in table1 averaged over 3 runs) are very encouraging: the mixture of trees is the absolute winner for compressing the simple digits and comes in second as a model for pairs of digits. This suggests that our model (just like the mixture of factorized distributions) is able to perform good compression of the digit data but is unable to discover the independency in the double digit set.

# 6  CONCLUSIONS

This paper has shown a method of modeling and exploiting sparse dependency structure that is conditioned on values of the data. By using trees, our method avoids the exponential computation demands that plague both inference and structure finding in wider classes of belief nets. The algorithms presented here are linear in $m$ and $N$ and quadratic in $|V|$. Each M step is performing exact maximization over the space of all the tree structures and parameters. The possibility of pruning the edges of the components of a mixture of trees can play a role in classification, as a means of automatically selecting the variables that are relevant for the task.

The importance of using the right priors in constructing models for real-world problems can hardly be understated. In this context, the present paper has presented a broad class of priors that are efficiently handled in the framework of our algorithm and it has shown that this class includes important priors like the MDL prior and the Dirichlet prior.

### Acknowledgements

Thanks to Quaid Morris for running the digits and structure finding experiments and to Brendan Frey for providing the digits datasets.

## Footnotes

[1]In the graph theory literature, our definition corresponds to a *forest*. The connected components of a forest are called trees.

[2] http://www.ics.uci.edu/~mlearn/MLRepository.html

# References

Chow, C. K. and Liu, C. N. (1968). Approximating discrete probability distributions with dependence trees. *"IEEE Transactions on Information Theory"*, IT-14(3):462–467.

Dempster, A. P., Laird, N. M., and Rubin, D. B. (1977). Maximum likelihood from incomplete data via the EM algorithm. *Journal of the Royal Statistical Society, B*, 39:1–38.

Frey, B. J., Hinton, G. E., and Dayan, P. (1996). Does the wake-sleep algorithm produce good density estimators? In Touretsky, D., Mozer, M., and Hasselmo, M., editors, *Neural Information Processing Systems*, number 8, pages 661–667. MIT Press.

Friedman, N. and Goldszmidt, M. (1996). Building classifiers using Bayesian networks. In *Proceedings of the National Conference on Artificial Intelligence (AAAI 96)*, pages 1277–1284, Menlo Park, CA. AAAI Press.

Geiger, D. (1992). An entropy-based learning algorithm of bayesian conditional trees. In *Proceedings of the 8th Conference on Uncertainty in AI*, pages 92–97. Morgan Kaufmann Publishers.

Geiger, D. (1996). Knowledge representation and inference in similarity networks and bayesian multinets. *"Artificial Intelligence"*, 82:45–74.

Heckerman, D., Geiger, D., and Chickering, D. M. (1995). Learning Bayesian networks: the combination of knowledge and statistical data. *Machine Learining*, 20(3):197–243.

Kontkanen, P., Myllymaki, P., and Tirri, H. (1996). Constructing bayesian finite mixture models by the EM algorithm. Technical Report C-1996-9, Univeristy of Helsinky. Department of Computer Science.

Meilă, M., Jordan, M. I., and Morris, Q. D. (1997). Estimating dependency structure as a hidden variable. Technical Report AIM–1611,CBCL–151, Massachusetts Institute of Technology, Artificial Intelligence Laboratory.

Pearl, J. (1988). *Probabilistic Reasoning in Intelligent Systems: Networks of Plausible Inference*. Morgan Kaufman Publishers, San Mateo, CA.

Thiesson, B., Meek, C., Chickering, D. M., and Heckerman, D. (1997). Learning mixtures of Bayes networks. Technical Report MSR–POR–97–30, Microsoft Research.